# Monte Carlo Value Iteration with Macro-Actions

**Zhan Wei Lim**       **David Hsu**       **Wee Sun Lee**

Department of Computer Science, National University of Singapore
Singapore, 117417, Singapore

## Abstract

POMDP planning faces two major computational challenges: large state spaces and long planning horizons. The recently introduced Monte Carlo Value Iteration (MCVI) can tackle POMDPs with very large discrete state spaces or continuous state spaces, but its performance degrades when faced with long planning horizons. This paper presents *Macro-MCVI*, which extends MCVI by exploiting macro-actions for temporal abstraction. We provide sufficient conditions for Macro-MCVI to inherit the good theoretical properties of MCVI. Macro-MCVI does not require explicit construction of probabilistic models for macro-actions and is thus easy to apply in practice. Experiments show that Macro-MCVI substantially improves the performance of MCVI with suitable macro-actions.

## 1 Introduction

Partially observable Markov decision process (POMDP) provides a principled general framework for planning with imperfect state information. In POMDP planning, we represent an agent's possible states probabilistically as a *belief* and systematically reason over the space of all beliefs in order to derive a policy that is robust under uncertainty. POMDP planning, however, faces two major computational challenges. The first is the "curse of dimensionality". A complex planning task involves a large number of states, which result in a high-dimensional belief space. The second obstacle is the "curse of history". In applications such as robot motion planning, an agent often takes many actions before reaching the goal, resulting in a long planning horizon. The complexity of the planning task grows very fast with the horizon.

Point-based approximate algorithms [10, 14, 9] have brought dramatic progress to POMDP planning. Some of the fastest ones, such as HSVI [14] and SARSOP [9], can solve moderately complex POMDPs with hundreds of thousands states in reasonable time. The recently introduced Monte Carlo Value Iteration (MCVI) [2] takes one step further. It can tackle POMDPs with very large discrete state spaces or continuous state spaces. The main idea of MCVI is to sample both an agent's state space and the corresponding belief space simultaneously, thus avoiding the prohibitive computational cost of unnecessarily processing these spaces in their entirety. It uses Monte Carlo sampling in conjunction with dynamic programming to compute a policy represented as a finite state controller. Both theoretical analysis and experiments on several robotic motion planning tasks indicate that MCVI is a promising approach for plannning under uncertainty with very large state spaces, and it has already been applied successfully to compute the threat resolution logic for aircraft collision avoidance systems in 3-D space [1].

However, the performance of MCVI degrades, as the planning horizon increases. Temporal abstraction using macro-actions is effective in mitigating the negative effect and has achieved good results in earlier work on Markov decision processes (MDPs) and POMDPs (see Section 2). In this work, we show that macro-actions can be seamlessly integrated into MCVI, leading to the *Macro-MCVI* algorithm. Unfortunately, the theoretical properties of MCVI, such as the approximation error bounds [2], do not carry over to Macro-MCVI automatically, if arbitrary mapping from belief to actions are allowed as macro-actions. We give sufficient conditions for the good theoretical properties

to be retained, tranforming POMDPs into a particular type of partially observable semi-Markov decision processes (POSMDPs) in which the lengths of macro-actions are not observable.

A major advantage of the new algorithm is its ability to abstract away the lengths of macro-actions in planning and reduce the effect of long planning horizons. Furthermore, it does not require explicit probabilistic models for macro-actions and treats them just like primitive actions in MCVI. This simplifies macro-action construction and is a major benefit in practice. Macro-MCVI can also be used to construct a hierarchy of macro-actions for planning large spaces. Experiments show that the algorithm is effective with suitably designed macro-actions.

## 2 Related Works

Macro-actions have long been used to speed up planning and learning algorithms for MDPs (see, *e.g.*, [6, 15, 3]). Similarly, they have been used in offline policy computation for POMDPs [16, 8]. Macro-actions can be composed hierarchically to further improve scalability [4, 11]. These earlier works rely on vector representations for beliefs and value functions, making it difficult to scale up to large state spaces. Macro-actions have also been used in online search algorithms for POMDPs [7].

Macro-MCVI is related to Hansen and Zhou's work [5]. The earlier work uses finite state controllers for policy representation and policy iteration for policy computation, but it has not yet been shown to work on large state spaces. Expectation-maximization (EM) can be used to train finite state controllers [17] and potentially handle large state spaces, but it often gets stuck in local optima.

## 3 Planning with Macro-actions

We would like to generalize POMDPs to handle macro-actions. Ideally, the generalization should retain properties of POMDPs such as piecewise linear and convex finite horizon value functions. We would also like the approximation bounds for MCVI [2] to hold with macro-actions.

We would like to allow our macro-actions to be as powerful as possible. A very powerful representation for a macro-action would be to allow it to be an arbitrary mapping from belief to action that will run until some termination condition is met. Unfortunately, the value function of a process with such macro-actions need not even be continuous. Consider the following simple finite horizon example, with horizon one. Assume that there are two primitive actions, both with constant rewards, regardless of state. Consider two macro-actions, one which selects the poorer primitive action all the time while the other which selects the better primitive action for some beliefs. Clearly, the second macro-action dominates the first macro-action over the entire belief space. The reward for the second macro-action takes two possible values depending on which action is selected for the belief. The reward function also forms the optimal value function of the process and need not even be continuous as the macro-action can be an arbitrary mapping from belief to action.

Next, we give sufficient conditions for the process to retain piecewise linearity and convexity of the value function. We do this by constructing a type of partially observable semi-Markov decision process (POSMDP) with the desired property. The POSMDP does not need to have the length of the macro-action observed, a property that can be practically very useful as it allows the branching factor for search to be significantly smaller. Furthermore, the process is a strict generalization of a POMDP as it reduces to a POMDP when all the macro-actions have length one.

### 3.1 Partially Observable Semi-Markov Decision Process

Finite-horizon (undiscounted) POSMDP were studied in [18]. Here, we focus on a type of infinite-horizon discounted POSMDPs whose transition intervals are not observable. Our POSMDP is formally defined as a tuple $(S, \mathcal{A}, \mathcal{O}, \mathbf{T}, \mathbf{R}, \gamma)$, where $S$ is a state space, $\mathcal{A}$ is a macro-action space, $\mathcal{O}$ is a macro-observation space, $\mathbf{T}$ is a joint transition and observation function, $\mathbf{R}$ is a reward function, and $\gamma \in (0, 1)$ is a discount factor. If we apply a macro-action $\mathbf{a}$ with start state $s_i$, $\mathbf{T} = p(s_j, \mathbf{o}, k | s_i, \mathbf{a})$ encodes the joint conditional probability of the end state $s_j$, macro-observation $\mathbf{o}$, and the number of time steps $k$ that it takes for $\mathbf{a}$ to reach $s_j$ from $s_i$. We could decompose $\mathbf{T}$ into a state-transition function and an observation function, but avoid doing so here to remain general and simplify the notation. The reward function $\mathbf{R}$ gives the discounted cumulative reward for a macro-action $\mathbf{a}$ that starts at state $s$: $\mathbf{R}(s, \mathbf{a}) = \sum_{t=0}^{\infty} \gamma^t \mathrm{E}(r_t | s, \mathbf{a})$, where $\mathrm{E}(r_t | s, \mathbf{a})$ is the expected reward at step $t$. Here we assume that the reward is 0 once a macro-action terminates.

For convenience, we will work with reweighted beliefs, instead of beliefs. Assuming that the number of states is $n$, a reweighted belief (like a belief) is a vector of $n$ non-negative numbers that sums to

one. By assuming that the POSMDP process will stop with probability $1-\gamma$ at each time step, we can interpret the reweighted belief as the conditional probability of a state given that the process has not stopped. This gives an interpretation of the reweighted belief in terms of the discount factor. Given a reweighted belief, we compute the next reweighted belief given macroaction $\mathbf{a}$ and observation $\mathbf{o}$, $b' = \tau(b, \mathbf{a}, \mathbf{o})$, as follows:

$$b'(s) = \frac{\sum_{k=1}^{\infty} \gamma^{k-1} \sum_{i=1}^{n} p(s, \mathbf{o}, k|s_i, \mathbf{a})b(s_i)}{\sum_{k=1}^{\infty} \gamma^{k-1} \sum_{j=0}^{n} \sum_{i=1}^{n} p(s_j, \mathbf{o}, k|s_i, \mathbf{a})b(s_i)}. \tag{1}$$

We will simply refer to the reweighted belief as a belief from here on. We denote the denominator $\sum_{k=1}^{\infty} \gamma^{k-1} \sum_{j=0}^{n} \sum_{i=1}^{n} p(s_j, \mathbf{o}, k|s_i, \mathbf{a})b(s_i)$ by $p_\gamma(\mathbf{o}|\mathbf{a}, b)$. The value of $\gamma p_\gamma(\mathbf{o}|\mathbf{a}, b)$ can be interpreted as the probability that observation $\mathbf{o}$ is received and the POSMDP has not stopped. Note that $\sum_{\mathbf{o}} p_\gamma(\mathbf{o}|\mathbf{a}, b)$ may sum to less than 1 due to discounting.

A policy $\pi$ is a mapping from a belief to a macro-action. Let $\mathbf{R}(b, \mathbf{a}) = \sum_s b(s)\mathbf{R}(s, \mathbf{a})$. The value of a policy $\pi$ can be defined recursively as

$$V_\pi(b) = \mathbf{R}(b, \pi(b)) + \gamma \sum_{\mathbf{o}} p_\gamma(\mathbf{o}|\pi(b), b) V_\pi(\tau(b, \pi(b), \mathbf{o})).$$

Note that the policy operates on the belief and may not know the number of steps taken by the macro-actions. If knowledge of the number of steps is important, it can be added into the observation function in the modeling process.

We now define the backup operator $H$ that operates on a value function $V_m$ and returns $V_{m+1}$

$$HV(b) = \max_{\mathbf{a}} \big( R(b, \mathbf{a}) + \gamma \sum_{\mathbf{o} \in \mathcal{O}} p_\gamma(\mathbf{o}|\mathbf{a}, b) V(\tau(b, \mathbf{a}, \mathbf{o})) \big). \tag{2}$$

The backup operator is a contractive mapping[1].

**Lemma 1** *Given value functions $U$ and $V$, $||HU - HV||_\infty \leq \gamma ||U - V||_\infty$.*

Let the value of an optimal policy, $\pi^*$, be $V^*$. The following theorem is a consequence of the Banach fixed point theorem and Lemma 1.

**Theorem 1** *$V^*$ is the unique fixed point of $H$ and satisfies the Bellman equation $V^* = HV^*$.*

We call a policy an $m$-step policy if the number of times the macro-actions is applied is $m$. For $m$-step policies, $V^*$ can be approximated by a finite set of linear functions; the weight vectors of these linear functions are called the $\alpha$-vectors.

**Theorem 2** *The value function for an $m$-step policy is piecewise linear and convex and can be represented as*

$$V_m(b) = \max_{\alpha \in \Gamma_m} \sum_{s \in S} \alpha(s)b(s) \tag{3}$$

*where $\Gamma_m$ is a finite collection of $\alpha$-vectors.*

As $V_m$ is convex and converges to $V^*$, $V^*$ is also convex.

### 3.2 Macro-action Construction

We would like to construct macro-actions from primitive actions of a POMDP in order to use temporal abstraction to help solve difficult POMDP problems. A partially observable Markov decision process (POMDP) is defined by finite state space $S$, finite action space $A$, a reward function $R(s, a)$, an observation space $O$, and a discount $\gamma \in (0, 1)$.

In our POSMDP, the probability function $p(s_j, \mathbf{o}, k|s_i, \mathbf{a})$ for a macro-action must be independent of the history given the current state $s_i$; hence the selection of primitive actions and termination conditions within the macro-action cannot depend on the belief. We examine some allowable dependencies here. Due to partial observability, it is often not possible to allow the primitive action and the termination condition to be functions of the initial state. Dependence on the portion of history

that occurs after the macro-action has started is, however, allowed. In some POMDPs, a subset of the state variables are always observed and can be used to decide the next action. In fact, we may sometimes explicitly construct observed variables to remember relevant parts of the history prior to the start of macro-action (see Section 5); these can be considered as parameters that are passed on to the macro-action. Hence, one way to construct the next action in a macro-action is to make it a function of the history since the macro-action started, $x_k, a_k, o_{k+1}, \ldots, x_{t-1}, a_{t-1}, o_t, x_t$, where $x_i$ is the fully observable subset of state variables at time $i$, and $k$ is the starting time of the macro-action.

Similarly, when the termination criterion and the observation function of the macro-action depends only on the history $x_k, a_k, o_{k+1}, \ldots, x_{t-1}, a_{t-1}, o_t, x_t$, the macro-action can retain a transition function that is independent of the history given the initial state. Note that the observation to be passed on to the POSMDP to create the POSMDP observation space, $\mathcal{O}$, is part of the design trade-off - usually it is desirable to reduce the number of observations in order to reduce complexity without degrading the value of the POSMDP too much. In particular, we may not wish to include the execution length of the macro-action if it does not contribute much towards obtaining a good policy.

## 4 Monte Carlo Value Iteration with Macro-Actions

We have shown that if the action space $\mathcal{A}$ and the observation space $\mathcal{O}$ of a POSMDP are discrete, then the optimal value function $V^*$ can be approximated arbitrarily closely by a piecewise-linear, convex function. Unfortunately, when $S$ is very high-dimensional (or continuous), a vector representation is no longer effective. In this section, we show how the Monte Carlo Value Iteration (MCVI) algorithm [2], which has been designed for POMDPs with very large or infinite state spaces, can be extended to POSMDP.

Instead of $\alpha$-vectors, MCVI uses an alternative policy representation called a *policy graph* $G$. A policy graph is a directed graph with labeled nodes and edges. Each node of $G$ is labeled with an macro-action $\mathbf{a}$ and each edge of $G$ is labeled with an observation $\mathbf{o}$. To execute a policy $\pi_G$, it is treated as a finite state controller whose states are the nodes of $G$. Given an initial belief $b$, a starting node $v$ of $G$ is selected and its associated macro-action $\mathbf{a}_v$ is performed. The controller then transitions from $v$ to a new node $v'$ by following the edge $(v, v')$ labeled with the observation received, $\mathbf{o}$. The process then repeats with the new controller node $v'$.

Let $\pi_{G,v}$ denote a policy represented by $G$, when the controller always starts in node $v$ of $G$. We define the value $\alpha_v(s)$ to be the expected total reward of executing $\pi_{G,v}$ with initial state $s$. Hence

$$V_G(b) = \max_{v \in G} \sum_{s \in S} \alpha_v(s)b(s). \tag{4}$$

$V_G$ is completely determined by the $\alpha$-functions associated with the nodes of $G$.

### 4.1 MC-Backup

One way to approximate the value function is to repeatedly run the backup operator $H$ starting from an arbitrary value function until it is close to convergence. This algorithm is called *value iteration* (VI). Value iteration can be carried out on policy graphs as well, as it provides an implicit representation of a value function. Let $V_G$ be the value function for a policy graph $G$. Substituting (4) into (2), we get

$$HV_G(b) = \max_{\mathbf{a} \in \mathcal{A}} \left\{ \sum_{s \in S} \mathbf{R}(s, a)b(s) + \sum_{\mathbf{o} \in \mathcal{O}} p_\gamma(\mathbf{o}|\mathbf{a}, b) \max_{v \in G} \sum_{s \in S} \alpha_v(s)b'(s) \right\}. \tag{5}$$

It is possible to then evaluate the right-hand side of (5) via sampling and monte carlo simulation at a belief $b$. The outcome is a new policy graph $G'$ with value function $\hat{H}_b V_G$. This is called MC-backup of $G$ at $b$ (Algorithm 1) [2].

There are $|\mathcal{A}||G|^{|O|}$ possible ways to generate a new policy graph $G'$ which has one new node compared to the old policy graph node. Algorithm 1 computes an estimate of the best new policy graph at $b$ using only $N|\mathcal{A}||G|$ samples. Furthermore, we can show that MC-backup approximates the standard VI backup (equation (5)) well at $b$, with error decreasing at the rate $O(1/\sqrt{N})$. Let $R_{\max}$ be the largest absolute value of the reward, $|r_t|$, at any time step.

**Algorithm 1** MC-Backup of a policy graph $G$ at a belief $b \in \mathcal{B}$ with $N$ samples.

MC-BACKUP$(G, b, N)$
1: For each action $\mathbf{a} \in \mathcal{A}$, $R_{\mathbf{a}} \leftarrow 0$.
2: For each action $\mathbf{a} \in \mathcal{A}$, each observation $\mathbf{o} \in \mathcal{O}$, and each node $v \in G$, $V_{\mathbf{a},\mathbf{o},v} \leftarrow 0$.
3: **for** each action $\mathbf{a} \in \mathcal{A}$ **do**
4:     **for** $i = 1$ to $N$ **do**
5:         Sample a state $s_i$ with probability $b(s_i)$.
6:         Simulate taking macro-action $\mathbf{a}$ in state $s_i$. Generate a new state $s_i'$, observation $\mathbf{o}_i$, and discounted reward $R'(s_i, \mathbf{a})$ by sampling from $p(s_j, \mathbf{o}, k | s_i, \mathbf{a})$.
7:         $R_{\mathbf{a}} \leftarrow R_{\mathbf{a}} + R'(s_i, \mathbf{a})$.
8:         **for** each node $v \in G$ **do**
9:             Set $V'$ to be the expected total reward of simulating the policy represented by $G$, with initial controller state $v$ and initial state $s_i'$.
10:             $V_{\mathbf{a},\mathbf{o}_i,v} \leftarrow V_{\mathbf{a},\mathbf{o}_i,v} + V'$.
11:     **for** each observation $\mathbf{o} \in \mathcal{O}$ **do**
12:         $V_{\mathbf{a},\mathbf{o}} \leftarrow \max_{v \in G} V_{\mathbf{a},\mathbf{o},v}$.
13:         $v_{\mathbf{a},\mathbf{o}} \leftarrow \operatorname{argmax}_{v \in G} V_{\mathbf{a},\mathbf{o},v}$.
14:     $V_{\mathbf{a}} \leftarrow (R_{\mathbf{a}} + \gamma \sum_{\mathbf{o} \in \mathcal{O}} V_{\mathbf{a},\mathbf{o}})/N$.
15: $V^* \leftarrow \max_{\mathbf{a} \in \mathcal{A}} V_{\mathbf{a}}$.
16: $\mathbf{a}^* \leftarrow \operatorname{argmax}_{\mathbf{a} \in \mathcal{A}} V_{\mathbf{a}}$.
17: Create a new policy graph $G'$ by adding a new node $u$ to $G$. Label $u$ with $\mathbf{a}^*$. For each $\mathbf{o} \in \mathcal{O}$, add the edge $(u, v_{\mathbf{a}^*,\mathbf{o}})$ and label it with $\mathbf{o}$.
18: **return** $G'$.

**Theorem 3** *Given a policy graph $G$ and a point $b \in B$, MC-BACKUP$(G, b, N)$ produces an improved policy graph such that*

$$|\hat{H}_b V_G(b) - H V_G(b)| \leq \frac{2R_{\max}}{1 - \gamma} \sqrt{\frac{2\big(|\mathcal{O}| \ln |G| + \ln(2|\mathcal{A}|) + \ln(1/\tau)\big)}{N}},$$

*with probability at least $1 - \tau$.*

The proof uses Hoeffding bound together with union bound. Details can be found in [2].

MC-backup can be combined with point-based POMDP planning, which samples the belief space $\mathcal{B}$. Point-based POMDP algorithms use a set $B$ of points sampled from $\mathcal{B}$ as an approximate representation of $\mathcal{B}$. In contrast to the standard VI backup operator $H$, which performs backup at every point in $\mathcal{B}$, the operator $\hat{H}_B$ applies MC-BACKUP$(G_m, b, N)$ on a policy graph $G_m$ at every point in $B$. This results in $|B|$ new policy graph nodes. $\hat{H}_B$ then produces a new policy graph $G_{m+1}$ by adding the new policy graph nodes to the previous policy graph $G_m$.

Let $\delta_B = \sup_{b \in \mathcal{B}} \min_{b' \in B} \|b - b'\|_1$ be the maximum $L_1$ distance from any point in $\mathcal{B}$ to the closest point in $B$. Let $V_0$ be value function for some initial policy graph and $V_{m+1} = \hat{H}_B V_m$. The theorem below bounds the approximation error between $V_m$ and the optimal value function $V^*$.

**Theorem 4** *For every $b \in B$,*

$$|V^*(b) - V_m(b)| \leq \frac{2R_{\max}}{(1 - \gamma)^2} \sqrt{\frac{2\big(|\mathcal{O}| \ln(|B|m) + \ln(2|\mathcal{A}|) + \ln(|B|m/\tau)\big)}{N}} + \frac{2R_{\max}}{(1 - \gamma)^2} \delta_B + \frac{2\gamma^m R_{\max}}{(1 - \gamma)},$$

*with probability at least $1 - \tau$.*

The proof requires the contraction property and a Lipschitz property that can be derived from the piece-wise linearity of the value function. Having established those results in Section 3.1, the rest of the proof follows from the proof in [2]. The first term in the bound in Theorem 4 comes from Theorem 3, showing that the error from sampling decays at the rate $O(1/\sqrt{N})$ and can be reduced by taking a large enough sample size. The second term depends on how well the set $B$ covers $\mathcal{B}$ and can be reduced by sampling a larger number of beliefs. The last term depends on the number of MC-backup iterations and decays exponentially with $m$.

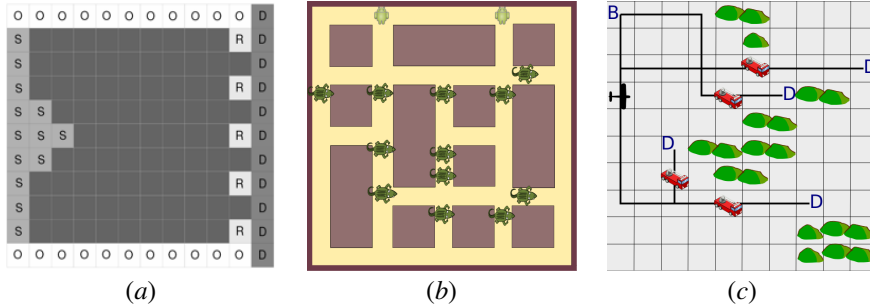

|     |     |     |
| :-: | :-: | :-: |
| (a) | (b) | (c) |

Figure 1: (a) Underwater Navigation: A reduced map with a $11 \times 12$ grid is shown with "S" marking the possible initial positions, "D" marking the destinations, "R" marking the rocks and "O" marking the locations where the robot can localize completely. (b) Collaborative search and capture: Two robotic agents catching 12 escaped crocodiles in a $21 \times 21$ grid. (c) Vehicular ad-hoc networking: An UAV maintains ad-hoc network over four ground vehicles in a $10 \times 10$ grid with "B" marking the base and "D" the destinations.

## 4.2  Algorithm

Theorem 4 bounds the performance of the algorithm when given a set of beliefs. Macro-MCVI, like MCVI, samples beliefs incrementally in practice and performs backup at the sampled beliefs. Branch and bound is used to avoid sampling unimportant parts of the belief space. See [2] for details.

The other important component in a practical algorithm is the generation of next belief; Macro-MCVI uses a particle filter for that. Given the macro-action construction as described in Section 3.2, a simple particle filter is easily implemented to approximate the next belief function in equation (1): sample a set of states from the current belief; from each sampled state, simulate the current macro-action until termination, keeping track of its path length, $t$; if the observation at termination matches the desired observation, keep the particle; the set of particles that are kept are weighted by $\gamma^t$ and then renormalized to form the next belief[2]. Similarly, MC-backup is performed by simply running simulations of the macro-actions - there is no need to store additional transition and observation matrices, allowing the method to run for very large state spaces.

## 5   Experiments

We now illustrate the use of macro-actions for temporal abstraction in three POMDPs of varying complexity. Their state spaces range from relatively small to very large. Correspondingly, the macro-actions range from relatively simple ones to much more complex ones forming a hierarchy.

**Underwater Navigation:** The underwater navigation task was introduced in [9]. In this task, an autonomous underwater vehicle (AUV) navigates in an environment modeled as 51 x 52 grid map. The AUV needs to move from the left border to the right border while avoiding the rocks scattered near its destination. The AUV has six actions: move north, move south, move east, move north-east, move south-east or stay in the same location. Due to poor visibility, the AUV can only localize itself along the top or bottom borders where there are beacon signals.

This problem has several interesting characteristics. First, the relatively small state space size of 2653 means that solvers that use $\alpha$-vectors, such as SARSOP [9] can be used. Second, the dynamics of the robot is actually noiseless, hence the main difficulty is actually localization from the robot's initially unknown location.

We use 5 macro-actions that move in a direction (north, south, east, north-east, or south-east) until either a beacon signal or the destination is reached. We also define an additional macro-action that: navigates to the nearest goal location if the AUV position is known, or simply stays in the same location if the AUV position is not known. To enable proper behaviour of the last macro-action, we augment the state space with a fully observable state variable that indicates the current AUV location. The variable is initialized to a value denoting "unknown" but takes the value of the current AUV location after the beacon signal is received. This gives a simple example where the original state space is augmented with a fully observable state variable to allow more sophisticated macro-action behaviour.

**Collaborative Search and Capture:** In this problem, a group of crocodiles had escaped from its enclosure into the environment and two robotic agents have to collaborate to hunt down and capture the crocodiles (see Figure 1). Both agents are centrally controlled and each agent can make a one step move in one of the four directions (north, south, east and west) or stay still at each time instance. There are twelve crocodiles in the environment. At every time instance, each crocodile moves to a location furthest from the agent that is nearest to it with a probability $1 - p$ ($p = 0.05$ in the experiments). With a probability $p$, the crocodile moves randomly. A crocodile is captured when it is at the same location as an agent. The agents do not know the exact location of the crocodiles, but each agent knows the number of crocodiles in the top left, top right, bottom left and bottom right quadrants around itself from the noise made by the crocodiles. Each captured crocodile gives a reward of 10, while movement is free.

We define twenty-five macro actions where each agent moves (north, south, east, west, or stay) along a passage way until one of them reaches an intersection. In addition, the macro-actions only return the observation it makes at the point when the macro-action terminates, reducing the complexity of the problem, possibly at a cost of some sub-optimality. In this problem, the macro-actions are simple, but the state space is extremely large (approximately $179^{14}$).

**Vehicular Ad-hoc Network:** In a post disaster search and rescue scenario, a group of rescue vehicles are deployed for operation work in an area where communication infrastructure has been destroyed. The rescue units need high-bandwidth network to relay images of ground situations. An Unmanned Aerial Vehicle (UAV) can be deployed to maintain WiFi network communication between the ground units. The UAV needs to visit each vehicle as often as possible to pick up and deliver data packets [13].

In this task, 4 rescue vehicles and 1 UAV navigates in a terrain modeled as a 10 x 10 grid map. There are obstacles on the terrain that are impassable to ground vehicle but passable to UAV. The UAV can move in one of the four directions (north, south, east, and west) or stay in the same location at every time step. The vehicles set off from the same base and move along some predefined path towards their pre-assigned destinations where they will start their operations, randomly stopping along the way. Upon reaching its destination, the vehicle may roam around the environment randomly while carrying out its mission. The UAV knows its own location on the map and can observe the location of a vehicle if they are in the same grid square. To elicit a policy with low network latency, there is a penalty of $-0.1\times$ number of time steps since last visit of a vehicle for each time step for each vehicle. There is a reward of 10 for each time a vehicle is visited by the UAV. The state space consists of the vehicles' locations, UAV location in the grid map and the number of time steps since each vehicle is last seen (for computing the reward).

We abstract the movements of UAV to search and visit a single vehicle as macro actions. There are two kinds of search macro actions for each vehicle: search for a vehicle along its predefined path and search for a vehicle that has started to roam randomly. To enable the macro-actions to work effectively, the state space is also augmented with the previous seen location of each vehicle. Each macro-action is in turn hierarchically constructed by solving the simplified POMDP task of searching for a single vehicle on the same map using basic actions and some simple macro-actions that move along the paths. This problem has both complex hierarchically constructed macro-actions and very large state space.

## 5.1 Experimental setup

We applied Macro-MCVI to the above tasks and compared its performance with the original MCVI algorithm. We also compared with a state-of-the-art off-line POMDP solver, SARSOP [9], on the underwater navigation task. SARSOP could not run on the other two tasks, due to their large state space sizes. For each task, we ran Macro-MCVI until the average total reward stablized. We then ran the competing algorithms for at least the same amount of time. The exact running times are difficult to control because of our implementation limitations. To confirm the comparison results, we also ran the competing algorithms 100 times longer when possible. All experiments were conducted on a 16 core Intel Xeon 2.4Ghz computer server.

Neither MCVI nor SARSOP uses macro-actions. We are not aware of other efficient off-line macro-action POMDP solvers that have been demonstrated on very large state space problems. Some online search algorithms, such as PUMA [7], use macro-actions and have shown strong results. Online search algorithms do not generate a policy, making a fair comparison difficult. Despite that, they

are useful as baseline references; we implement a variant of PUMA as a one such reference. In our experiments, we simply gave the online search algorithms as much or more time than Macro-MCVI and report the results here. PUMA uses open-loop macro-actions. As a baseline reference for online solvers with closed-loop macro-actions, we also created an online search variant of Macro-MCVI by removing the MC-backup component. We refer to this variant as *Online-Macro*. It is similar to other recent online POMDP algorithms [12], but uses the same closed-loop macro-actions as MCVI does.

## 5.2 Results

The performance of the different algorithms is shown in Figure 2 with 95% confidence intervals.

The underwater navigation task consist of two phases: the localization phase and navigate to goal phase. Macro-MCVI's policy takes one macro-action, "moving northeast until reaching the border", to localize and another macro-action, "navigating to the goal", to reach the goal. In contrast, both MCVI and SARSOP fail to match the performance of Macro-MCVI even when they are run 100 times longer. Online-Macro does well, as the planning horizon is short with the use of macro-actions. PUMA, however, does not do as well, as it uses the less powerful open-loop macro-actions, which move in the same direction for a fixed number of time steps.

For the collaborative search & capture task, MCVI fails to match the performance of Macro-MCVI even when it is run for 100 times longer. PUMA and Online-Macro do badly as they fail to search deep

Figure 2: Performance comparison.

|  | Reward | Time(s) |
|---|---|---|
| **Underwater Navigation** | | |
| Macro-MCVI | 749.30 ± 0.28 | 1 |
| MCVI | 678.05 ± 0.48 | 4 |
|  | 725.28 ± 0.38 | 100 |
| SARSOP | 710.71 ± 4.52 | 1 |
|  | 730.83 ± 0.75 | 100 |
| PUMA | 697.47 ± 4.58 | 1 |
| Online-Macro | 746.10 ± 2.37 | 1 |
| **Collaborative Search & Capture** | | |
| Macro-MCVI | 17.04 ± 0.03 | 120 |
| MCVI | 13.14 ± 0.04 | 120 |
|  | 16.38 ± 0.05 | 12000 |
| PUMA | 1.04 ± 0.91 | 144 |
| Online-Macro | 0 | 3657 |
| **Vehicular Ad-Hoc Network** | | |
| Macro-MCVI | -323.55 ± 3.79 | 29255 |
| MCVI | -1232.57 ± 2.24 | 29300 |
| Greedy | -422.26 ± 3.98 | 28800 |

enough and do not have the benefit of reusing sub-policies obtained from the backup operation. To confirm that it is the backup operation and not the shorter per macro-action time that is responsible for the performance difference, we ran Online-Macro for a much longer time and found the result unchanged.

The vehicular ad-hoc network task was solved hierarchically in two stages. We first used Macro-MCVI to solve for the policy that finds a single vehicle. This stage took roughly 8 hours of computation time. We then used the single-vehicle policy as a macro-action and solved for the higher-level policy that plans over the macro-actions. Although it took substantial computation time, Macro-MCVI generated a reasonable policy in the end. In contrast, MCVI, without macro-actions, fails badly for this task. Due to the long running time involved, we did not run MCVI 100 times longer. To confirm that that the policy computed by Macro-MCVI at the higher level of the hierarchy is also effective, we manually crafted a greedy policy over the single-vehicle macro-actions. This greedy policy always searches for the vehicle that has not been visited for the longest duration. The experimental results indicate that the higher-level policy computed by Macro-MCVI is more effective than the greedy policy. We did not apply online algorithms to this task, as we are not aware of any simple way to hierarchically construct macro-actions online.

## 6 Conclusions

We have successfully extended MCVI, an algorithm for solving very large state space POMDPs, to include macro-actions. This allows MCVI to use temporal abstraction to help solve difficult POMDP problems. The method inherits the good theoretical properties of MCVI and is easy to apply in practice. Experiments show that it can substantially improve the performance of MCVI when used with appropriately chosen macro-actions.

**Acknowledgement** We thank Tomás Lozano-Pérez and Leslie Kaelbling from MIT for many insightful discussions. This work is supported in part by MoE grant MOE2010-T2-2-071 and MDA GAMBIT grant R-252-000-398-490.

## Footnotes

[1]Proofs of the results in this section are included in the supplementary material.

[2]More sophisticated approximation of the belief can be constructed but may require more knowledge of the underlying POMDP and more computation.

# References

[1] H. Bai, D. Hsu, M.J. Kochenderfer, and W. S. Lee. Unmanned aircraft collision avoidance using continuous-state POMDPs. In *Proc. Robotics: Science & Systems*, 2011.

[2] H. Bai, D. Hsu, W. S. Lee, and V. Ngo. Monte Carlo Value Iteration for Continuous-State POMDPs. In *Algorithmic Foundations of Robotics IX—Proc. Int. Workshop o n the Algorithmic Foundations of Robotics (WAFR)*, pages 175–191. Springer, 2011.

[3] Andrew G. Barto and Sridhar Mahadevan. Recent advances in hierarchical reinforcement learning. *Discrete Event Dynamic Systems*, 13:2003, 2003.

[4] T. G. Dietterich. Hierarchical reinforcement learning with the MAXQ value function decomposition. *J. Artificial Intelligence Research*, 13:227–303, 2000.

[5] E. Hansen and R. Zhou. Synthesis of hierarchical finite-state controllers for POMDPs. In *Proc. Int. Conf. on Automated Planning and Scheduling*, 2003.

[6] M. Hauskrecht, N. Meuleau, L.P. Kaelbling, T. Dean, and C. Boutilier. Hierarchical solution of Markov decision processes using macro-actions. In *Proc. Conf. on Uncertainty in Artificial Intelligence*, pages 220–229. Citeseer, 1998.

[7] R. He, E. Brunskill, and N. Roy. PUMA: Planning under uncertainty with macro-actions. In *Proc. AAAI Conf. on Artificial Intelligence*, 2010.

[8] H. Kurniawati, Y. Du, D. Hsu, and W. S. Lee. Motion planning under uncertainty for robotic tasks with long time horizons. *Int. J. Robotics Research*, 30(3):308–323, 2010.

[9] H. Kurniawati, D. Hsu, and W.S. Lee. SARSOP: Efficient point-based POMDP planning by approximating optimally reachable belief spaces. In *Proc. Robotics: Science & Systems*, 2008.

[10] J. Pineau, G. Gordon, and S. Thrun. Point-based value iteration: An anytime algorithm for POMDPs. In *Int. Jnt. Conf. on Artificial Intelligence*, volume 18, pages 1025–1032, 2003.

[11] J. Pineau, N. Roy, and S. Thrun. A hierarchical approach to POMDP planning and execution. In *Workshop on Hierarchy & Memory in Reinforcement Learning (ICML)*, volume 156, 2001.

[12] S. Ross, J. Pineau, S. Paquet, and B. Chaib-Draa. Online planning algorithms for POMDPs. *Journal of Artificial Intelligence Research*, 32(1):663–704, 2008.

[13] A. Sivakumar and C.K.Y. Tan. UAV swarm coordination using cooperative control for establishing a wireless communications backbone. In *Proc. Int. Conf. on Autonomous Agents & Multiagent Systems*, pages 1157–1164, 2010.

[14] T. Smith and R. Simmons. Heuristic search value iteration for POMDPs. In *Proc. Conf. on Uncertainty in Artificial Intelligence*, pages 520–527. AUAI Press, 2004.

[15] R.S. Sutton, D. Precup, and S. Singh. Between MDPs and semi-MDPs: A framework for temporal abstraction in reinforcement learning. *Artificial Intelligence*, 112(1):181–211, 1999.

[16] G. Theocharous and L. P. Kaelbling. Approximate planning in POMDPs with macro-actions. *Advances in Neural Processing Information Systems*, 17, 2003.

[17] M. Toussaint, L. Charlin, and P. Poupart. Hierarchical POMDP controller optimization by likelihood maximization. *Proc. Conf. on Uncertainty in Artificial Intelligence*, 2008.

[18] C.C. White. Procedures for the solution of a finite-horizon, partially observed, semi-Markov optimization problem. *Operations Research*, 24(2):348–358, 1976.

